# An Improved Policy Iteration Algorithm for Partially Observable MDPs

**Eric A. Hansen**
Computer Science Department
University of Massachusetts
Amherst, MA 01003
hansen@cs.umass.edu

## Abstract

A new policy iteration algorithm for partially observable Markov decision processes is presented that is simpler and more efficient than an earlier policy iteration algorithm of Sondik (1971,1978). The key simplification is representation of a policy as a finite-state controller. This representation makes policy evaluation straightforward. The paper's contribution is to show that the dynamic-programming update used in the policy improvement step can be interpreted as the transformation of a finite-state controller into an improved finite-state controller. The new algorithm consistently outperforms value iteration as an approach to solving infinite-horizon problems.

## 1   Introduction

A partially observable Markov decision process (POMDP) is a generalization of the standard completely observable Markov decision process that allows imperfect information about the state of the system. First studied as a model of decision-making in operations research, it has recently been used as a framework for decision-theoretic planning and reinforcement learning with hidden state (Monahan, 1982; Cassandra, Kaelbling, & Littman, 1994; Jaakkola, Singh, & Jordan, 1995).

Value iteration and policy iteration algorithms for POMDPs were first developed by Sondik and rely on a piecewise linear and convex representation of the value function (Sondik, 1971; Smallwood & Sondik,1973; Sondik, 1978). Sondik's policy iteration algorithm has proved to be impractical, however, because its policy evaluation step is extremely complicated and difficult to implement. As a result, almost all subsequent work on dynamic programming for POMDPs has used value iteration. In this paper, we describe an improved policy iteration algorithm for POMDPs that avoids the difficulties of Sondik's algorithm. We show that these difficulties hinge on the choice of a policy representation and can be avoided by representing a policy as a finite-state

controller. This representation makes the policy evaluation step easy to implement and efficient. We show that the policy improvement step can be interpreted in a natural way as the transformation of a finite-state controller into an improved finite-state controller. Although it is not always possible to represent an optimal policy for an infinite-horizon POMDP as a finite-state controller, it is always possible to do so when the optimal value function is piecewise linear and convex. Therefore representation of a policy as a finite-state controller is no more limiting than representation of the value function as piecewise linear and convex. In fact, it is the close relationship between representation of a policy as a finite-state controller and representation of a value function as piecewise linear and convex that the new algorithm successfully exploits.

The paper is organized as follows. Section 2 briefly reviews the POMDP model and Sondik's policy iteration algorithm. Section 3 describes an improved policy iteration algorithm. Section 4 illustrates the algorithm with a simple example and reports a comparison of its performance to value iteration. The paper concludes with a discussion of the significance of this work.

## 2 Background

Consider a discrete-time POMDP with a finite set of states $S$, a finite set of actions $A$, and a finite set of observations $\Theta$. Each time period, the system is in some state $i \in S$, an agent chooses an action $a \in A$ for which it receives a reward with expected value $r_i^a$, the system makes a transition to state $j \in S$ with probability $p_{ij}^a$, and the agent observes $\theta \in \Theta$ with probability $q_{j\theta}^a$. We assume the performance objective is to maximize expected total discounted reward over an infinite horizon.

Although the state of the system cannot be directly observed, the probability that it is in a given state can be calculated. Let $\pi$ denote a vector of state probabilities, called an *information state*, where $\pi_i$ denotes the probability that the system is in state $i$. If action $a$ is taken in information state $\pi$ and $\theta$ is observed, the successor information state is determined by revising each state probability using Bayes' theorem: $\pi_j = \sum_{i \in S} \pi_i p_{ij}^a q_{j\theta}^a / \sum_{i,j \in S} \pi_i p_{ij}^a q_{j\theta}^a$. Geometrically, each information state $\pi$ is a point in the $(|S| - 1)$-dimensional unit simplex, denoted $\Pi$.

It is well-known that an information state $\pi$ is a sufficient statistic that summarizes all information about the history of a POMDP necessary for optimal action selection. Therefore a POMDP can be recast as a completely observable MDP with a continuous state space $\Pi$ and it can be theoretically solved using dynamic programming. The key to practical implementation of a dynamic-programming algorithm is a piecewise-linear and convex representation of the value function. Smallwood and Sondik (1973) show that the dynamic-programming update for POMDPs preserves the piecewise linearity and convexity of the value function. They also show that an optimal value function for a finite-horizon POMDP is always piecewise linear and convex. For infinite-horizon POMDPs, Sondik (1978) shows that an optimal value function is sometimes piecewise linear and convex and can be approximated arbitrarily closely by a piecewise linear and convex function otherwise.

A piecewise linear and convex value function $V$ can be represented by a finite set of $|S|$-dimensional vectors, $\Gamma = \{\alpha^0, \alpha^1, \ldots\}$, such that $V(\pi) = \max_k \sum_{i \in S} \pi_i \alpha_i^k$. A dynamic-programming update transforms a value function $V$ represented by a set $\Gamma$ of $\alpha$-vectors into an improved value function $V'$ represented by a set $\Gamma'$ of $\alpha$-vectors. Each possible $\alpha$-vector in $\Gamma'$ corresponds to choice of an action, and for each possible observation, choice of a successor vector in $\Gamma$. Given the combinatorial number of choices that can be made, the maximum number of vectors in $\Gamma'$ is $|A||\Gamma|^{|\Theta|}$. However most of these potential vectors are not needed to define the updated value function and can be pruned. Thus the dynamic-programming update problem is to find a

minimal set of vectors $\Gamma'$ that represents $V'$, given a set of vectors $\Gamma$ that represents $V$. Several algorithms for performing this dynamic-programming update have been developed but describing them is beyond the scope of this paper. Any algorithm for performing the dynamic-programming update can be used in the policy improvement step of policy iteration. The algorithm that is presently the fastest is described by (Cassandra, Littman, & Zhang, 1997).

For value iteration, it is sufficient to have a representation of the value function because a policy is defined implicitly by the value function, as follows,

$$\delta(\pi) = a(\arg\max_k \sum_{i \in S} \pi_i \alpha_i^k), \tag{1}$$

where $a(k)$ denotes the action associated with vector $\alpha^k$. But for policy iteration, a policy must be represented independently of the value function because the policy evaluation step computes the value function of a given policy. Sondik's choice of a policy representation is influenced by Blackwell's proof that for a continuous-space infinite-horizon MDP, there is a stationary, deterministic Markov policy that is optimal (Blackwell, 1965). Based on this result, Sondik restricts policy space to stationary and deterministic Markov policies that map the continuum of information space $\Pi$ into action space $A$. Because it is important for a policy to have a finite representation, Sondik defines an admissible policy as a mapping from a finite number of polyhedral regions of $\Pi$ to $A$. Each region is represented by a set of linear inequalities, where each linear inequality corresponds to a boundary of the region.

This is Sondik's canonical representation of a policy, but his policy iteration algorithm makes use of two other representations. In the policy evaluation step, he converts a policy from this representation to an equivalent, or approximately equivalent, finite-state controller. Although no method is known for computing the value function of a policy represented as a mapping from $\Pi$ to $A$, the value function of a finite-state controller can be computed in a straightforward way. In the policy improvement step, Sondik converts a policy represented implicitly by the updated value function and equation (1) back to his canonical representation. The complexity of translating between these different policy representations – especially in the policy evaluation step – makes Sondik's policy iteration algorithm difficult to implement and explains why it is not used in practice.

## 3   Algorithm

We now show that policy iteration for POMDPs can be simplified – both conceptually and computationally – by using a single representation of a policy as a finite-state controller.

### 3.1   Policy evaluation

As Sondik recognized, policy evaluation is straightforward when a policy is represented as a finite-state controller. An $\alpha$-vector representation of the value function of a finite-state controller is computed by solving the system of linear equations,

$$\alpha_i^k = r_i^{a(k)} + \beta \sum_{j,\theta} p_{ij}^{a(k)} q_{j\theta}^{a(k)} \alpha_j^{s(k,\theta)}, \tag{2}$$

where $k$ is an index of a state of the finite-state controller, $a(k)$ is the action associated with machine state $k$, and $s(k,\theta)$ is the index of the successor machine state if $\theta$ is observed. This value function is convex as well as piecewise linear because the expected value of an information state is determined by assuming the controller is started in the machine state that optimizes it.

---

1. Specify an initial finite-state controller, $\delta$, and select $\epsilon$ for detecting convergence to an $\epsilon$-optimal policy.

2. Policy evaluation: Calculate a set $\Gamma$ of $\alpha$-vectors that represents the value function for $\delta$ by solving the system of equations given by equation 2.

3. Policy improvement: Perform a dynamic-programming update and use the new set of vectors $\Gamma'$ to transform $\delta$ into a new finite-state controller, $\delta'$, as follows:

   (a) For each vector $\alpha$ in $\Gamma'$:
      i. If the action and successor links associated with $\alpha$ duplicate those of a machine state of $\delta$, then *keep* that machine state unchanged in $\delta'$.
      ii. Else if $\alpha$ pointwise dominates a vector associated with a machine state of $\delta$, *change* the action and successor links of that machine state to those used to create $\alpha$. (If it pointwise dominates the vectors of more than one machine state, they can be combined into a single machine state.)
      iii. Otherwise *add* a machine state to $\delta'$ that has the same action and successor links used to create $\alpha$.

   (b) *Prune* any machine state for which there is no corresponding vector in $\Gamma'$, as long as it is not reachable from a machine state to which a vector in $\Gamma'$ does correspond.

4. Termination test. If the Bellman residual is less than or equal to $\epsilon(1 - \beta)/\beta$, exit with $\epsilon$-optimal policy. Otherwise set $\delta$ to $\delta'$ and go to step 2.

Figure 1: Policy iteration algorithm.

## 3.2  Policy improvement

The policy improvement step uses the dynamic-programming update to transform a value function $V$ represented by a set $\Gamma$ of $\alpha$-vectors into an improved value function $V'$ represented by a set $\Gamma'$ of $\alpha$-vectors. We now show that the dynamic-programming update can also be interpreted as the transformation of a finite-state controller $\delta$ into an improved finite-state controller $\delta'$. The transformation is made based on a simple comparison of $\Gamma'$ and $\Gamma$.

First note that some of the $\alpha$-vectors in $\Gamma'$ are duplicates of $\alpha$-vectors in $\Gamma$, that is, their action and successor links match (and their vector values are pointwise equal). Any machine state of $\delta$ for which there is a duplicate vector in $\Gamma'$ is left unchanged. The vectors in $\Gamma'$ that are not duplicates of vectors in $\Gamma$ indicate how to change the finite-state controller. If a non-duplicate vector in $\Gamma'$ pointwise dominates a vector in $\Gamma$, the machine state that corresponds to the pointwise dominated vector in $\Gamma$ is changed so that its action and successor links match those of the dominating vector in $\Gamma'$. If a non-duplicate vector in $\Gamma'$ does not pointwise dominate a vector in $\Gamma$, a machine state is added to the finite-state controller with the same action and successor links used to generate the vector. There may be some machine states for which there is no corresponding vector in $\Gamma'$ and they can be pruned, but only if they are not reachable from a machine state that corresponds to a vector in $\Gamma'$. This last point is important because it preserves the integrity of the finite-state controller.

A policy iteration algorithm that uses these simple transformations to change a finite-state controller in the policy improvement step is summarized in Figure 1. An algorithm that performs this transformation is easy to implement and runs very efficiently because it simply compares the $\alpha$-vectors in $\Gamma'$ to the $\alpha$-vectors in $\Gamma$ and modifies the finite-state controller accordingly. The policy evaluation step is invoked to compute the value function of the transformed finite-state controller. (This is only necessary

if a machine state has been changed, not if machine states have simply been added.) It is easy to show that the value function of the transformed finite-state controller $\delta'$ dominates the value function of the original finite-state controller, $\delta$, and we omit the proof which appears in (Hansen, 1998).

**Theorem 1** *If a finite-state controller is not optimal, policy improvement transforms it into a finite-state controller with a value function that is as good or better for every information state and better for some information state.*

### 3.3 Convergence

If a finite-state controller cannot be improved in the policy improvement step (i.e., all the vectors in $\Gamma'$ are duplicates of vectors in $\Gamma$), it must be optimal because the value function satisfies the optimality equation. However policy iteration does not necessarily converge to an optimal finite-state controller after a finite number of iterations because there is not necessarily an optimal finite-state controller. Therefore we use the same stopping condition used by Sondik to detect $\epsilon$-optimality: a finite-state controller is $\epsilon$-optimal when the Bellman residual is less than or equal to $\epsilon(1-\beta)/\beta$, where $\beta$ denotes the discount factor. Representation of a policy as a finite-state controller makes the following proof straightforward (Hansen, 1998).

**Theorem 2** *Policy iteration converges to an $\epsilon$-optimal finite-state controller after a finite number of iterations.*

## 4 Example and performance

We illustrate the algorithm using the same example used by Sondik: a simple two-state, two-action, two-observation POMDP that models the problem of finding an optimal marketing strategy given imperfect information about consumer preferences (Sondik,1971,1978). The two states of the problem represent consumer preference or lack of preference for the manufacturers brand; let $B$ denote brand preference and $\neg B$ denote lack of brand preference. Although consumer preferences cannot be observed, they can be infered based on observed purchasing behavior; let $P$ denote purchase of the product and let $\neg P$ denote no purchase. There are two marketing alternatives or actions; the company can market a luxury version of the product (L) or a standard version (S). The luxury version is more expensive to market but can bring greater profit. Marketing the luxury version also increases brand preference. However consumers are more likely to purchase the less expensive, standard product. The transition probabilities, observation probabilities, and reward function for this example are shown in Figure 2. The discount factor is 0.9.

Both Sondik's policy iteration algorithm and the new policy iteration algorithm converge in three iterations from a starting policy that is equivalent to the finite-state

| Actions | Transition probabilities | | Observation probabilities | | Expected reward | |
|---|---|---|---|---|---|---|
| | **B** | **−B** | **P** | **−P** | | |
| Market luxury product (L) | **B** 0.8 | 0.2 | **B** 0.8 | 0.2 | **B** | 4 |
| | **−B** 0.5 | 0.5 | **−B** 0.6 | 0.4 | **−B** | -4 |
| | **B** | **−B** | **P** | **−P** | | |
| Market standard product (S) | **B** 0.5 | 0.5 | **B** 0.9 | 0.1 | **B** | 0 |
| | **−B** 0.4 | 0.6 | **−B** 0.4 | 0.6 | **−B** | -3 |

Figure 2: Parameters for marketing example of Sondik (1971,1978).

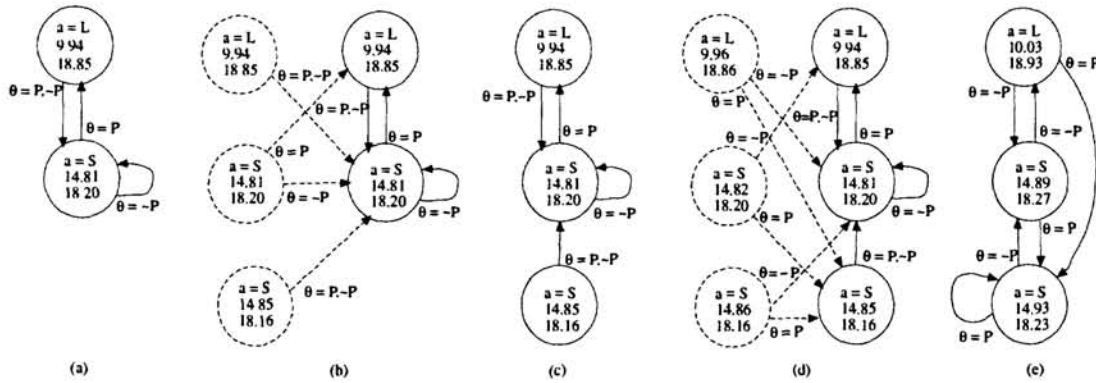

Figure 3: (a) shows the initial finite-state controller, (b) uses dashed circles to show the vectors in $\Gamma'$ generated in the first policy improvement step and (c) shows the transformed finite-state controller, (d) uses dashed circles to show the vectors in $\Gamma'$ generated in the second policy improvement step and (e) shows the transformed finite-state controller after policy evaluation. The optimality of this finite-state controller is detected on the third iteration, which is not shown. Arcs are labeled with one of two possible observations and machine states are labeled with one of two possible actions and a 2-dimensional vector that contains a value for each of the two possible system states.

controller shown in Figure 3a. Figure 3 shows how the initial finite-state controller is transformed into an optimal finite-state controller by the new algorithm. In the first iteration, the updated set of vectors $\Gamma'$ (indicated by dashed circles in Figure 3b) includes two duplicate vectors and one non-duplicate that results in an added machine state. Figure 3c shows the improved finite-state controller after the first iteration. In the second iteration, each of the three vectors in the updated set of vectors $\Gamma'$ (indicated by dashed circles in Figure 3d) pointwise dominates a vector that corresponds to a current machine state. Thus each of these machine states is changed. Figure 4e shows the improved finite-state controller after the second iteration. The optimality of this finite-state controller is detected in the third iteration.

This is the only example for which Sondik reports using policy iteration to find an optimal policy. For POMDPs with more than two states, Sondik's algorithm is especially difficult to implement. Sondik reports that his algorithm finds a suboptimal policy for an example described in (Smallwood & Sondik, 1973). No further computational experience with his algorithm has been reported.

The new policy iteration algorithm described in this paper easily finds an optimal finite-state controller for the example described in (Smallwood & Sondik, 1973) and has been used to solve many other POMDPs. In fact, it consistently outperforms value iteration. We compared its performance to the performance of value iteration on a suite of ten POMDPs that represent a range of problem sizes for which exact dynamic-programming updates are currently feasible. (Presently, exact dynamic-prorgramming updates are not feasible for POMDPs with more than about ten or fifteen states, actions, or observations.) Starting from the same point, we measured how soon each algorithm converged to $\epsilon$-optimality for $\epsilon$ values of 10.0, 1.0, 0.1, and 0.01. Policy iteration was consistently faster than value iteration by a factor that ranged from a low of about 10 times faster to a high of over 120 times faster. On average, its rate of convergence was between 40 and 50 times faster than value iteration for this set of examples. The finite-state controllers it found had as many as several hundred machine states, although optimal finite-state controllers were sometimes found with just a few machine states.

# 5 Discussion

We have demonstrated that the dynamic-programming update for POMDPs can be interpreted as the improvement of a finite-state controller. This interpretation can be applied to both value iteration and policy iteration. It provides no computational speedup for value iteration, but for policy iteration it results in substantial speedup by making policy evaluation straightforward and easy to implement. This representation also has the advantage that it makes a policy easier to understand and execute than representation as a mapping from regions of information space to actions. In particular, a policy can be executed without maintaining an information state at run-time.

It is well-known that policy iteration converges to $\epsilon$-optimality (or optimality) in fewer iterations than value iteration. For completely observable MDPs, this is not a clear advantage because the policy evaluation step is more computationally expensive than the dynamic-programming update. But for POMDPs, policy evaluation has low-order polynomial complexity compared to the worst-case exponential complexity of the dynamic-programming update (Littman et al., 1995). Therefore, policy iteration appears to have a clearer advantage over value iteration for POMDPs. Preliminary testing bears this out and suggests that policy iteration significantly outperforms value iteration as an approach to solving infinite-horizon POMDPs.

### Acknowledgements

Thanks to Shlomo Zilberstein and especially Michael Littman for helpful discussions. Support for this work was provided in part by the National Science Foundation under grants IRI-9409827 and IRI-9624992.

## References

Blackwell, D. (1965) Discounted dynamic programming. *Ann. Math. Stat.* 36:226-235.

Cassandra, A.; Kaelbling, L.P.; Littman, M.L. (1994) Acting optimally in partially observable stochastic domains. In *Proc. 13th National Conf. on AI*, 1023-1028.

Cassandra, A.; Littman, M.L.; & Zhang, N.L. (1997) Incremental pruning: A simple, fast, exact algorithm for partially observable Markov decision processes. In *Proc. 13th Annual Conf. on Uncertainty in AI*.

Hansen, E.A. (1998). *Finite-Memory Control of Partially Observable Systems*. PhD thesis, Department of Computer Science, University of Massachusetts at Amherst.

Jaakkola, T.; Singh, S.P.; & Jordan, M.I. (1995) Reinforcement learning algorithm for partially observable Markov decision problems. In NIPS-7.

Littman, M.L.; Cassandra, A.R.; & Kaebling, L.P. (1995) Efficient dynamic-programming updates in partially observable Markov decision processes. Computer Science Technical Report CS-95-19, Brown University.

Monahan, G.E. (1982) A survey of partially observable Markov decision processes: Theory, models, and algorithms. *Management Science* 28:1-16.

Smallwood, R.D. & Sondik, E.J. (1973) The optimal control of partially observable Markov processes over a finite horizon. *Operations Research* 21:1071-1088.

Sondik, E.J. (1971) *The Optimal Control of Partially Observable Markov Processes*. PhD thesis, Department of Electrical Engineering, Stanford University.

Sondik, E.J. (1978) The optimal control of partially observable Markov processes over the infinite horizon: Discounted costs. *Operations Research* 26:282-304.